# A Model of the Neural Basis of the Rat's Sense of Direction

**William E. Skaggs**
*bill@nsma.arizona.edu*

**James J. Knierim**
*jim@nsma.arizona.edu*

**Hemant S. Kudrimoti**
*hemant@nsma.arizona.edu*

**Bruce L. McNaughton**
*bruce@nsma.arizona.edu*
ARL Division of Neural Systems, Memory, And Aging
344 Life Sciences North, University of Arizona, Tucson AZ 85724

## Abstract

In the last decade the outlines of the neural structures subserving the sense of direction have begun to emerge. Several investigations have shed light on the effects of vestibular input and visual input on the head direction representation. In this paper, a model is formulated of the neural mechanisms underlying the head direction system. The model is built out of simple ingredients, depending on nothing more complicated than connectional specificity, attractor dynamics, Hebbian learning, and sigmoidal nonlinearities, but it behaves in a sophisticated way and is consistent with most of the observed properties of real head direction cells. In addition it makes a number of predictions that ought to be testable by reasonably straightforward experiments.

## 1 Head Direction Cells in the Rat

There is quite a bit of behavioral evidence for an intrinsic sense of direction in many species of mammals, including rats and humans (e.g., Gallistel, 1990). The first specific information regarding the neural basis of this "sense" came with the discovery by Ranck (1984) of a population of "head direction" cells in the dorsal presubiculum (also known as the "postsubiculum") of the rat. A head direction cell

fires at a high rate if and only if the rat's head is oriented in a specific direction. Many things could potentially cause a cell to fire in a head-direction dependent manner: what made the postsubicular cells particularly interesting was that when their directionality was tested with the rat at different locations, the head directions corresponding to maximal firing were consistently parallel, within the experimental resolution. This is difficult to explain with a simple sensory-based mechanism; it implies something more sophisticated.[1]

The postsubicular head direction cells were studied in depth by Taube *et al.* (1990a,b), and, more recently, head direction cells have also been found in other parts of the rat brain, in particular the anterior nuclei of the thalamus (Mizumori and Williams, 1993) and the retrosplenial (posterior cingulate) cortex (Chen *et al.*, 1994a,b). Interestingly, all of these areas are intimately associated with the hippocampal formation, which in the rat contains large numbers of "place" cells. Thus, the brain contains separate but neighboring populations of cells coding for location and cells coding for direction, which taken together represent much of the information needed for navigation.

Figure 1 shows directional tuning curves for three typical head direction cells from the anterior thalamus. In each of them the breadth of tuning is on the order of 90 degrees. This value is also typical for head direction cells in the postsubiculum and retrosplenial cortex, though in each of the three areas individual cells may show considerable variability.

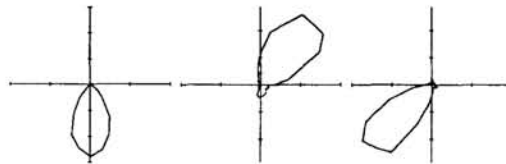

Figure 1: Polar plots of directional tuning (mean firing rate as a function of head direction) for three typical head direction cells from the anterior thalamus of a rat.

Every study to date has indicated that the head direction cells constitute a unitary system, together with the place cells of the hippocampus. Whenever two head direction cells have been recorded simultaneously, any manipulation that caused one of them to shift its directional alignment caused the other to shift by the same amount; and when head direction cells have been recorded simultaneously with place cells, any manipulation that caused the head direction cells to realign either caused the hippocampal place fields to rotate correspondingly or to "remap" into a different pattern (Knierim *et al.*, 1995).

Head direction cells maintain their directional tuning for some time when the lights in the recording room are turned off, leaving an animal in complete darkness; the directionality tends to gradually drift, though, especially if the animal moves around (Mizumori and Williams, 1993). Directional tuning is preserved to some degree even

if an animal is passively rotated in the dark, which indicates strongly that the head direction system receives information (possibly indirect) from the vestibular system.

Visual input influences but does not dictate the behavior of head direction cells. The nature of this influence is quite interesting. In a recent series of experiments (Knierim *et al.*, 1995), rats were trained to forage for food pellets in a gray cylinder with a single salient directional cue, a white card covering 90 degrees of the wall. During training, half of the rats were disoriented before being placed in the cylinder, in order to disrupt the relation between their internal sense of direction and the location of the cue card; the other half of the rats were not disoriented. Presumably, the rats that were not disoriented during training experienced the same initial relationship between their internal direction sense and the cue card each time they were placed in the cylinder; this would not have been true of the disoriented rats. Head direction cells in the thalamus were subsequently recorded from both groups of rats as they moved in the cylinder. *All rats were disoriented before each recording session.* Under these conditions, the cue card had much weaker control over the head direction cells in the rats that had been disoriented during training than in the rats that had not been disoriented. For all rats the influence of the cue card upon the head direction system weakened gradually over the course of multiple recording sessions, and eventually they broke free, but this happened much sooner in the rats that had been disoriented during training. The authors concluded that *a visual cue could only develop a strong influence upon the head direction system if the rat experienced it as stable.*

Figure 2 illustrates the shifts in alignment during a typical recording session. When the rat is initially placed in the cylinder, the cell's tuning curve is aligned to the west. Over the first few minutes of recording it gradually rotates to SSW, and there it stays. Note the "tail" of the curve. This comes from spikes belonging to another, neighboring head direction cell, which could not be perfectly isolated from the first. Note that, even though they come from different cells, both portions shift alignment synchronously.

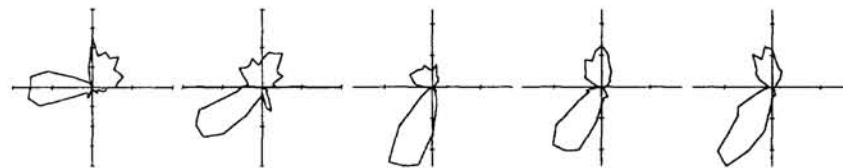

Figure 2: Shifts in alignment of a head direction cell over the course of a single recording session (one minute intervals).

## 2   The Model

As reviewed above, the most important facts to be accounted for by any model of the head direction system are (1) the shape of the tuning curves for head direction cells, (2) the control of head direction cells by vestibular input, and (3) the stability-dependent influence of visual cues on head direction cells. We introduce here a

model that accounts for these facts. It is a refinement of a model proposed earlier by McNaughton *et al.* (1991), the main addition being a more specific account of neural connections and dynamics. The aim of this effort is to develop the simplest possible architecture consistent with the available data. The reality is sure to be more complicated than this model.

Figure 3 schematically illustrates the architecture of the model. There are four groups of cells in the model: head direction cells, rotation cells (left and right), vestibular cells (left and right), and visual feature detectors. For expository purposes it is helpful to think of the network as a set of circular layers; this does not reflect the anatomical organization of the corresponding cells in the brain.

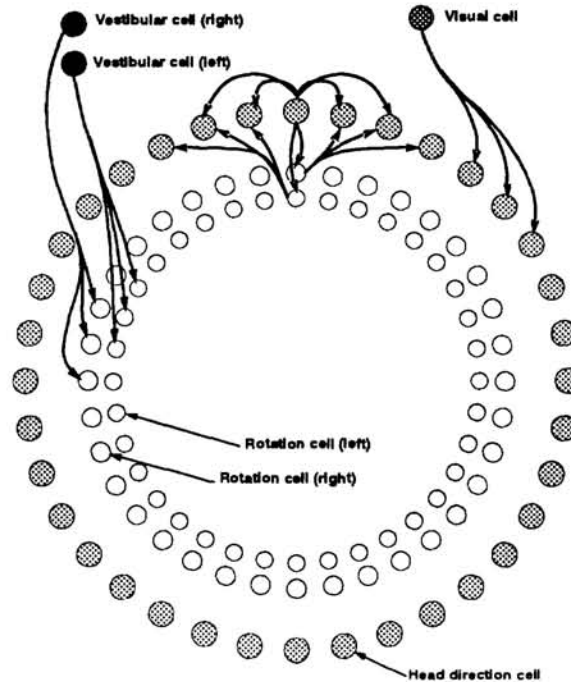

Figure 3: Architecture of the head direction cell model.

The head direction cell group has intrinsic connections that are stronger than any other connections in the model, and dominate their dynamics, so that other inputs only provide relatively small perturbations. The connections between them are set up so that the only possible stable state of the system is a single localized cluster of active cells, with all other cells virtually silent. This will occur if there are strong excitatory connections between neighboring cells, and strong inhibitory connections between distant cells. It is assumed that the network of interconnections has rotation and reflection symmetry. Small deviations from symmetry will not impair the model too much; large deviations may cause it to have strong attractors at a few points on the circle, which would cause problems.

The crucial property of this network is the following. Suppose it is in a stable state, with a single cluster of activated cells at one point on the circle, and suppose an external input is applied that excites the cells selectively on one side (left or right)

of the peak. Then the peak will rotate toward the side at which the input is applied, and the rate of rotation will increase with the strength of the input.

This feature is exploited by the mechanisms for vestibular and visual control of the system. The vestibular mechanism operates via a layer of "rotation" cells, corresponding to the circle of head direction cells (Units with a similar role were referred to as "$\mathbf{H} \times \mathbf{H}'$" cells in the McNaughton *et al.* (1991) model). There are two groups of rotation cells, for left and right rotations. Each rotation cell receives excitatory input from the head direction cell at the same point on the circle, and from the vestibular system. The activation function of the rotation cell is sigmoidal or threshold linear, so that the cell does not become active unless it receives input simultaneously from both sources. Each right rotation cell sends excitatory projections to head direction cells neighboring it on the right, but not to those that neighbor it on the left, and contrariwise for left rotation cells.

It is easy to see how the mechanism works. When the animal turns to the right, the right vestibular cells are activated, and then the right rotation cells at the current peak of the head direction system are activated. These add to the excitation of the head direction cells to the right of the peak, thereby causing the peak to shift rightward. This in turn causes a new set of rotation cells to become active (and the old ones inactive), and thence a further shift of the peak, and so on. The peak will continue to move around the circle as long as the vestibular input is active, and the stronger the vestibular input, the more rapidly the peak will move. If the gain of this mechanism is correct (but weak compared to the gain of the intrinsic connections of the head direction cells), then the peak will move around the circle at the same rate that the animal turns, and the location of the peak will function as an allocentric compass. This can only be expected to work over a limited range of turning rates, but the firing rates of cells in the vestibular nuclei are linearly proportional to angular velocity over a surprisingly broad range, so there is no reason why the mechanism cannot perform adequately.

Of course the mechanism is intrinsically error-prone, and without some sort of external correction, deviations are sure to build up over time. But this is an inevitable feature of any plausible model, and in any case does not conflict with the available data, which, while sketchy, suggests that passive rotation of animals in the dark can cause quite erratic behavior in head direction cells (E. J. Markus, J. J. Knierim, unpublished observations).

The final ingredient of the model is a set of visual feature detectors, each of which responds if and only if a particular visual feature is located at a particular angle with respect to the axis of the rat's head. Thus, these cells are feature specific and direction specific, but direction specific in the head-centered frame, not in the world frame. It is assumed that each visual feature detector projects weakly to all of the head direction cells, and that these connections are modifiable according to a Hebbian rule, specifically,

$$\Delta W = \alpha(W_{\max} f(\lambda_{\text{post}}) - W)\lambda_{\text{pre}},$$

where $W$ is the connection weight, $W_{\max}$ is its maximum possible value, $\lambda_{\text{post}}$ is the firing rate of the postsynaptic cell, $\lambda_{\text{pre}}$ is the firing rate of the presynaptic cell, and the function $f()$ has the shape shown in figure 4. (Actually, the rule is modified slightly to prevent any of the weights from becoming negative.) The net effect of

this rule is that the weight will only change when the presynaptic cell (the visual feature detector) is active, and the weight will increase if the postsynaptic cell is strongly active, but decrease if it is weakly active or silent. Modification rules of this form have previously been proposed in theories of the development of topography in the neocortex (e.g., Bienenstock *et al.*, 1982), and there is considerable evidence for such an effect in the control of LTP/LTD (Singer and Artola, 1994).

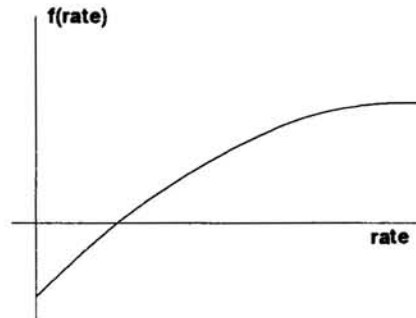

Figure 4: Dependence of synaptic weight change on postsynaptic firing rate for connections from visual feature detectors to head direction cells in the model.

To understand how this works, suppose we have a feature detecting cell that responds to a cue card whenever the cue card is directly in front of the rat. Suppose the rat's motion is restricted to a small area, and the cue card is far away, so that it is always at approximately the same absolute bearing (say, 30 degrees), and suppose the rat's head direction system is working correctly, i.e., functioning as an absolute compass. Then the cell will only be active at moments when the head direction cells corresponding to 30 degrees are active, and the Hebbian learning process will cause the feature detecting cell to be linked by strong weights to these cells, but by vanishing weights to other head direction cells. If the absolute bearing of the cue card were more variable, then the connection strengths from the feature detecting cell would be weaker and more broadly dispersed. In the limit where the bearing of the cue card was completely random, all connections would be weak and equal. Thus the influence of a visual cue is determined by the amount of training and by the variability in its bearing (with respect to the head direction system).

It can be seen that the model implements a competition between visual inputs and vestibular inputs for control of the head direction cells. If the visual cues are rotated while the rat is left stationary, then the head direction cells may either rotate to follow the visual cues, or stick with the inertial frame, depending on parameter values and, importantly, on the training regimen imposed on the network. Both of these outcomes have been observed in anterior thalamic head direction cells (McNaughton *et al.*, 1993).

## 3   Discussion

*Do the necessary types of cells exist in the brain?* Cells in the brainstem vestibular nuclei are known to have the properties required by the model (Precht, 1978). The "rotation" cells would be recognizable from the fact that they would fire only when

the rat is facing in a particular direction and turning in a particular direction, with rate at least roughly proportional to the speed of turning. Cells with these properties have been recorded in the postsubiculum (Markus *et al.*, 1990) and retrosplenial cortex (Chen *et al.*, 1994a). The visual cells would be recognizeable from the fact that they would respond to visual stimuli only when they come from a particular direction with respect to the animal's head axis. Cells with these properties have been recorded in the inferior parietal cortex, the internal medullary lamina of the thalamus, and the superior colliculus (e.g., Sparks, 1986). The superior colliculus also contains cells that respond in a direction-dependent manner to auditory inputs, thus allowing a possiblility of control of the head direction system by sound sources. There do not seem to be any strong direct projections from the superior colliculus to the components of the head direction system, but there are numerous indirect pathways.

The most general prediction of the model is that the influence of vestibular input upon head direction cells is not susceptible to experience-dependent modification, whereas the influence of visual input is plastic, and is enhanced by the duration of experience, the richness of the visual cue array, and the distance of visual cues from the rat's region of travel.

The "rotation" cells should be responsive to stimulation of the vestibular system. It is possible to activate the vestibular system by applying hot or cold water to the ears: if this is done in the dark, and head direction cells are simultaneously recorded, the model predicts that they will show periodic bursts of activity, with a frequency related to the intensity of the stimulus.

For another prediction, suppose we train two groups of rats to forage in a cylinder containing a single landmark. For one group, the landmark is placed at the edge of the cylinder; for the other group, the same landmark is placed halfway between the center and the edge. The model predicts that in both cases the landmark will influence the head direction sytem, but the influence will be stronger and more tightly focused when the landmark is at the edge.

In some respects the model is flexible, and may be extended without compromising its essence. For example, there is no intrinsic necessity that the vestibular system be the sole input to the rotation cells (other than the head direction cells). The performance of the system might be improved in some ways by sending the rotation cells input about optokinetic flow, or certain types of motor efference copy. But there is as yet no clear evidence for these things.

On a more abstract level, the mechanism used by the model for vestibular control may be thought of as a special case of a general-purpose method for integration with neurons. As such, it has significant advantages over some previously proposed neural integrators, in particular, better stability properties. It might be worth considering whether the method is applicable in other situations where integrators are known to exist, for example the control of eye position.

**Supported by MH46823 and O.N.R.**

## Footnotes

[1]Sensitivity to the Earth's geomagnetic field has been ruled out as an explanation of head-directional firing.

# References

Bienenstock, E. L., Cooper, L. N., and Munro, P. W. (1982). Theory for the development of neuron selectivity: orientation specificity and binocular interaction in visual cortex. *J. Neurosci.*, **2**:32–48.

Chen, L. L., Lin, L., Green, E. J., Barnes, C. A., and McNaughton, B. L. (1994b). Head-direction cells in the rat posterior cortex. I. Anatomical distribution and behavioral modulation. *Exp. Brain Res.*, **101**:8–23.

Chen, L. L., Lin, L., Barnes, C. A., and McNaughton, B. L. (1994a). Head-direction cells in the rat posterior cortex. II. Contributions of visual and ideothetic information to the directional firing. *Exp. Brain Res.*, **101**:24–34.

Gallistel, C. R. (1990). *The Organization of Learning*. MIT Press, Cambridge, Massachusetts.

Knierim, J. J., Kudrimoti, H. S., and McNaughton, B. L. (1995). Place cells, head direction cells, and the learning of landmark stability. *J. Neurosci.* (in press).

Markus, E. J., McNaughton, B. L., Barnes, C. A., Green, J. C., and Meltzer, J. (1990). Head direction cells in the dorsal presubiculum integrate both visual and angular velocity information. *Soc. Neurosci. Abs.*, **16**:441.

McNaughton, B. L., Chen, L. L., and Markus, E. J. (1991). "Dead reckoning," landmark learning, and the sense of direction: A neurophysiological and computational hypothesis. *J. Cognit. Neurosci.*, **3**:190–202.

McNaughton, B. L., Markus, E. J., Wilson, M. A., and Knierim, J. J. (1993). Familiar landmarks can correct for cumulative error in the inertially based dead-reckoning system. *Soc. Neurosci. Abs.*, **19**:795.

Mizumori, S. J. and Williams, J. D. (1993). Directionally selective mnemonic properties of neurons in the lateral dorsal nucleus of the thalamus of rats. *J. Neurosci.*, **13**:4015–4028.

Precht, W. (1978). *Neuronal operations in the vestibular system*. Springer, New York.

Ranck, Jr., J. B. (1984). Head direction cells in the deep cell layer of dorsal presubiculum in freely moving rats. *Soc. Neurosci. Abs.*, **10**:599.

Singer, W. and Artola, A. (1994). Plasticity of the mature neocortex. In Selverston, A. I. and Ascher, P., editors, *Cellular and molecular mechanisms underlying higher neural functions*, pages 49–69. Wiley.

Sparks, D. L. (1986). Translation of sensory signals into commands for control of saccadic eye movements: role of primate superior colliculus. *Physiol. Rev.*, **66**:118–171.

Taube, J. S., Muller, R. U., and Ranck, Jr., J. B. (1990a). Head direction cells recorded from the postsubiculum in freely moving rats. I. Description and quantitative analysis. *J. Neurosci.*, **10**:420–435.

Taube, J. S., Muller, R. U., and Ranck, Jr., J. B. (1990b). Head direction cells recorded from the postsubiculum in freely moving rats. II. Effects of environmental manipulations. *J. Neurosci.*, **10**:436–447.
